# Pattern Recognition from One Example by Chopping

**François Fleuret**
CVLAB/LCN – EPFL
Lausanne, Switzerland
`francois.fleuret@epfl.ch`

**Gilles Blanchard**[*]
Fraunhofer FIRST
Berlin, Germany
`blanchar@first.fhg.de`

## Abstract

We investigate the learning of the appearance of an object from a single image of it. Instead of using a large number of pictures of the object to recognize, we use a labeled reference database of pictures of other objects to learn invariance to noise and variations in pose and illumination. This acquired knowledge is then used to predict if two pictures of new objects, which do not appear on the training pictures, actually display the same object.

We propose a generic scheme called *chopping* to address this task. It relies on hundreds of random binary splits of the training set chosen to keep together the images of any given object. Those splits are extended to the complete image space with a simple learning algorithm. Given two images, the responses of the split predictors are combined with a Bayesian rule into a posterior probability of similarity.

Experiments with the COIL-100 database and with a database of 150 degraded LaTeX symbols compare our method to a classical learning with several examples of the positive class and to a direct learning of the similarity.

## 1 Introduction

Pattern recognition has so far mainly focused on the following task: given many training examples labelled with their classes (the object they display), guess the class of a new sample which was not available during training. The various approaches all consist of going to some invariant feature space, and there using a classification method such as neural networks, decision trees, kernel techniques, Bayesian estimations based on parametric density models, etc. Providing a large number of examples results in good statistical estimates of the model parameters. Although such approaches have been successful in applications to many problems, their performance are still far from what biological visual systems can do, which is *one sample learning*. This can be defined as the ability, given one picture of an object, to spot instances of the same object, under the assumption that these new views can be induced by the single available example.

---

[*]Supported in part by the IST Programme of the European Community, under the PASCAL Network of Excellence, IST-2002-506778

Being able to perform that type of one-sample learning corresponds to the ability, given one example, to sort out which elements of a test set are of the same class (i.e. one class vs. the rest of the world). This can be done by comparing one by one all the elements of the test set with the reference example, and labelling as of the same class those which are *similar enough*. Learning techniques can be used to choose the similarity measure, which could be adaptive and learned from a large number of examples of classes not involved in the test.

Thus, given a large number of training images of a large number of objects labeled with their actual classes, and provided two pictures of unknown objects (objects which *do not appear in the training pictures*), we want to decide if these two objects are actually the same object. The first image of such a couple can be seen as a single training example, and the second image as a test example. Averaging the error rate by repeating that test several times provides with an estimate of a one-sample learning (OSL) error rate.

The idea of "learning how to learn" is not new and has been applied in various settings [12]. Taking into account and/or learning relevant geometric invariances for a given task has been studied under various forms [1, 8, 11], and in [7] with the goal to achieve learning from very few examples. Finally, the precise one-sample learning setting considered here has been the object of recent research [4, 3, 5] proposing different methods (hyperfeature learning, distance learning) for finding invariant features from a set of training reference objects distinct from the test objects. This principle has also been dubbed *interclass transfer*.

The present study proposes a generic approach, and avoids an explicit description of the space of deformations. We propose to build a large number of binary *splits* of the image space, designed to assign the same binary label to all the images common to a same object. The binary mapping associated to such a split is thus highly invariant across the images of a certain object while highly variant across images of different objects. We can define such a split on the training images, and train a predictor to extend it to the complete image space by induction. We expect the predictor to respond similarly on two images of a same object, and differently on two images of two different objects with probability $\frac{1}{2}$. The global criterion to compare two images consists roughly of counting how many such split-predictors responds similarly and compare the result to a fixed threshold.

The principle of transforming a multiclass learning problem into several binary ones by class grouping has a long history in Machine Learning [10]. From this point of view the collected output of several binary classifiers is used as a way for coding class membership. In [2] it was proposed to carefully choose the class groupings so as to yield optimal separation of codewords (ECOC methodology). While our method is related to this general principle, our goal is different since we are interested in recognizing yet-unseen objects. Hence, the goal is not to code multiclass membership; our focus is not on designing efficient codes – splits are chosen randomly and we take a large number of them – but rather on how to use the learned mappings for learning unknown objects.

## 2   Data and features

To make the rest of the paper clearer to the reader, we now introduce the data and feature sets we are using for our proof of concept experiments. However, note that while we have focused on image classification, our approach is generic and could be applied to any signals for which adaptive binary classifiers are available.

### 2.1   Data

We use two databases of pictures for our experiments. The first one is the standard COIL-100 database of pictures [9]. It contains 7200 images corresponding to 100 different objects

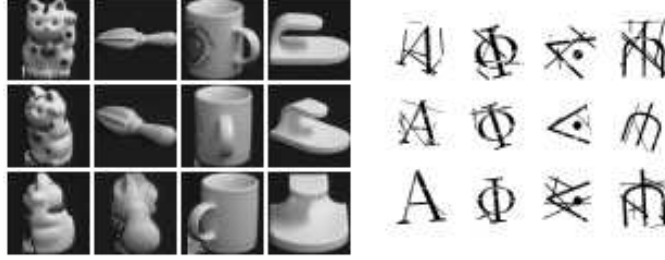

Figure 1: Four objects from the 100 objects of the COIL-100 database (downsampled to $38 \times 38$ grayscale pixels) and four symbols from the 150 symbols of our LaTeX symbol database ($A$, $\Phi$, $\lessdot$ and ⋔, resolution $28 \times 28$). Each image of the later is generated by applying a rotation and a scaling, and by adding lines of random grayscales at random locations and orientations.

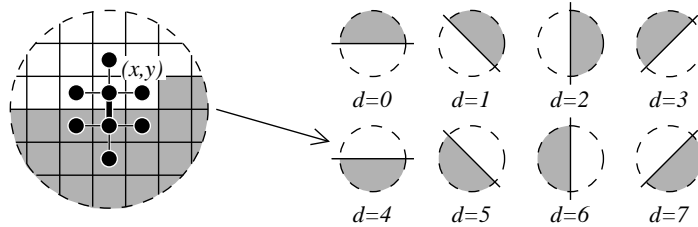

Figure 2: The figure on the left shows how an horizontal edge $\xi_{x,y,4}$ is detected: the six differences between pixels connected by a thin segment have to be all smaller in absolute value than the difference between the pixels connected by the thick segment. The relative values of the two pixels connected by the thick segment define the polarity of the edge (dark to light or light to dark). On the right are shown the eight different types of edges.

seen from 72 angles of view. We down-sample these images from their original resolution to $38 \times 38$ pixels, and convert them to grayscale. Examples are given in figure 1 (left). The second database contains images of 150 LaTeX symbols. We generated $1,000$ images of each symbol by applying a random rotation (angle is taken between $-20$ and $+20$ degrees) and a random scaling factor (up to 1.25). Noise is then added by adding random line segments of various gray scales, locations and orientations. The final resulting database contains $150,000$ images. Examples of these degraded images are given in figure 1 (right).

## 2.2 Features

All the classification processes in the rest of the paper are based on edge-based boolean features. Let $\xi_{x,y,d}$ denote a basic edge detector indexed by a location $(x,y)$ in the image frame and an orientation $d$ which can take eight different values, corresponding to four orientations and two polarities (see figure 2). Such an edge detector is equal to 1 if and only if an edge of the given location is detected at the specified location, and 0 otherwise. A feature $f_{x_0,y_0,x_1,y_1,d}$ is a disjunction of the $\xi$'s in the rectangle defined by $x_0, y_0, x_1, y_1$. Thus, it is equal to one if and only if $\exists x, y, x_0 \leq x \leq x_1, y_0 \leq y \leq y_1, \xi_{x,y,d} = 1$. For pictures of size $32 \times 32$ there is a total of $N = \frac{1}{4}(32 \times 32)^2 \times 8 \simeq 2.10^6$ features.

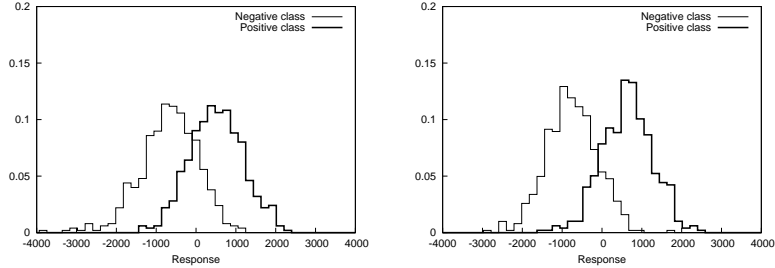

Figure 3: These two histograms are representative of the responses of two split predictors conditionally to the real arbitrary labelling $P(L \mid S)$.

## 3 Chopping

The main idea we propose in this paper consists of learning a large number of binary splits of the image space which would ideally assign the same binary label to all the images of any given object. In this section we define these splits and describe and justify how they are combined into a global rule.

### 3.1 Splits

A split is a binary labelling of the image space, with the property to give the same label to all images of a given object. We can trivially produce a labelling with that property on the training examples, but we need to be able to extend it to images not appearing in the training data, including images of other objects. We suppose that it is possible to infer a relevant split function on the complete image space, including images of other objects by looking at the problem as a binary classification problem. Inference is done by the mean of a simple learning scheme: a combination of a fast feature selection based on conditional mutual information (CMIM) [6] and a linear perceptron.

Thus, we create $M$ arbitrary splits on the training sample by randomly assigning the label 1 to half of the $N_T$ objects appearing in the training set, and 0 to the others. Since there are $\binom{N_T}{N_T/2}$ such balanced arbitrary labellings, with $N_T$ of the order of a few tens, a very large number of splits is available and only a small subset of them will be actually used for learning. For each one of those splits, we train a predictor using the scheme described above. Let $(S_1, \ldots, S_M)$ denote the family of arbitrary splits and $(L_1, \ldots, L_M)$ the split-predictors. The continuous outputs of these predictors before thresholding will be combined in the final classification.

### 3.2 Combining splits

To combine the responses of the various split predictors, we rely on a set of simple conditional independence assumptions (comparable to the "naive Bayes" setting) on the distribution of the true class label $C$ (each class corresponds to an object), the split labels $(S_i)$ and the predictor outputs $(L_i)$ for a single image. We do not assume that for test image pairs $(I_1, I_2)$ the two images are independent, because we want to encompass the case where pairs of images of the same object are much more frequent than they would be if they were independent (typically in our test data we have arranged to have $50\%$ of test pairs picturing the same object). We however still need some *conditional* independence assumption for the drawing of test image pairs. To simplify the notation we denote $L^1 = (L_i^1), L^2 = (L_i^2)$ the collection of predictor outputs for images 1 and 2, $S^1 = (S_i^1), S^2 = (S_i^2)$ the collection of their split labels and $C_1, C_2$ their true classes. The conditional indepence

assumptions we make are summed up in the following Markov dependency diagram:

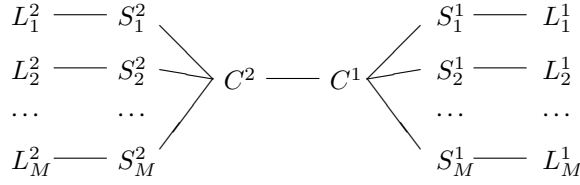

In words, for each split $i$, the predictor output $L_i$ is assumed to be independent of the true class $C$ conditionally to the split label $S_i$; and conditionally to the split labels $(S_1, S_2)$ of both images, the outputs of predictors on test pair images are assumed to be independent.

Finally, we make the additional symmetry hypothesis that conditionally to $C_1 = C_2$, for all $i : S_i^1 = S_i^2 = S_i$ and $(S_i)$ are independent Bernoulli variables with parameter $0.5$, while conditionally to $C_1 \neq C_2$ all split labels $(S_i^1, S_i^2)$ are independent Bernoulli(0.5).

Under these assumptions we then want to compute the log-odds ratio

$$\log \frac{P(C_1 = C_2 \mid L^1, L^2)}{P(C_1 \neq C_2 \mid L^1, L^2)} = \log \frac{P(L^1, L^2 \mid C_1 = C_2)}{P(L^1, L^2 \mid C_1 \neq C_2)} + \log \frac{P(C_1 = C_2)}{P(C_1 \neq C_2)} . \quad (1)$$

In this formula and the next ones, when handling real-valued variables $L_1, L_2$ we are implicitly assuming that they have a density with respect to the Lebesgue measure and probabilities are to be interpreted as densities with some abuse of notation. We assume that the second term above is either known or can be reliably estimated. For the first term, under the aforementioned independence assumptions, the following holds (see appendix):

$$\log \frac{P(L^1, L^2 \mid C_1 = C_2)}{P(L^1, L^2 \mid C_1 \neq C_2)} = N \log 2 + \sum_i \log \left( \alpha_i^1 \alpha_i^2 + (1 - \alpha_i^1)(1 - \alpha_i^2) \right) , \quad (2)$$

where $\alpha_i^j = P(S_i^j = 1 \mid L_i^j)$. As a quick check, note that if the predictor outputs $(L_i)$ are uninformative (i.e. every probability $\alpha_i^j$ is 0.5), then the above formula gives a ratio of 1 which is what we expect. If they are perfectly informative (i.e. all $\alpha_i^j$ are 0 or 1), the odds ratio can take the values 0 (if for some $j$ we can ensure $S_j^1 \neq S_j^2$, this excludes the case $C_1 = C_2$) or $2^N$ (if for all $j$ we have $S_j^1 = S_j^2$ there is still a tiny chance that $C_1 \neq C_2$ if by chance $C_1, C_2$ are on the same side of each split).

To estimate the probabilities $P(S_j \mid L_j)$, we use a simple 1D Gaussian model for the output of the predictor given the true split label. Mean and variance are estimated from the training set for each predictor. Experimental findings show that this Gaussian modelling is realistic (see figure 3).

## 4 Experiments

We estimate the performance of the chopping approach by comparing it to classical learning with several examples of the positive class and to a direct learning of the similarity of two objects on different images. For every experiment, we use a family of $10,000$ features sampled uniformly in the complete set of features (see section 2.2)

### 4.1 Multiple example learning

In this procedure, we train a predictor with several pictures of a positive class and with a very large number of pictures of a negative class. The number of positive examples depends on the experiments (from 1 to 32) and the number of negative examples is $2,000$

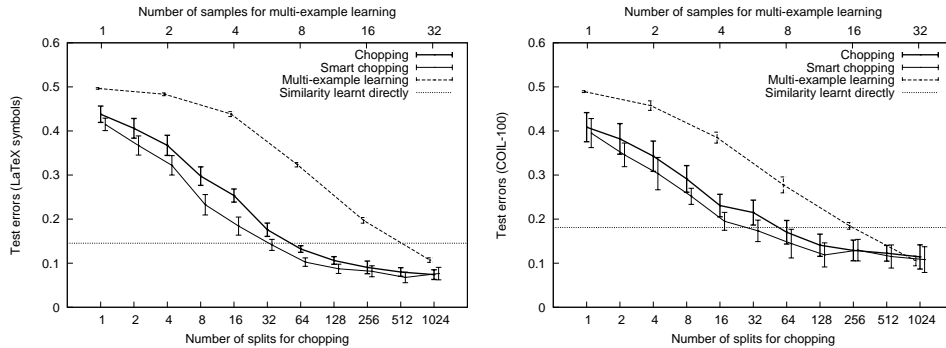

Figure 4: Error rates of the chopping, smart-chopping (see §4.2), multi-example learning and learnt similarity on the LaTeX symbol (left) and the COIL-100 database (right). Each curve shows the average error and a two standard deviation interval, both estimated on ten experiments for each setting. The $x$-axis shows either the number of splits for chopping or the number of samples of the positive class for the multi-example learning.

for both the COIL-100 and the LaTeX symbol databases. Note that to handle the unbalanced positive and negative populations, the perceptron bias is chosen to minimize a balanced error rate. In each case, and for each number of positive samples, we run 10 experiments. Each experiment consists of several cross-validation cycles so that the total number of test pictures is roughly the same as the number of pairs in one-sample techniques experiments below.

## 4.2 One-sample learning

For each experiment, whatever the predictor is, we first select 80 training objects from the COIL-100 database (respectively 100 symbols from the LaTeX symbol database). The test error is computed with 500 pairs of images of the 20 unseen objects for the COIL-100, and 1,000 pairs of images of the 50 unseen objects for the LaTeX symbols. These test sets are built to have as many pairs of images of the same object than pairs of images of different objects.

**Learnt similarity:** Note that one-sample learning can also be simply cast as a standard binary classification problem of pairs of images into the classes {*same, different*}. We therefore want to compare the Chopping method to a more standard learning method directly on pairs of images using a comparable set of features. For every single feature $f$ on single images, we consider three features of a pair of images standing for the conjunction, disjunction and equality of the feature responses on the two images. From the 10,000 features on single images, we thus create a set of 30,000 features on pairs of images.

We generate a training set of 2,000 pairs of pictures for the experiments with the COIL-100 database and 5,000 for the LaTeX symbols, half picturing the same object twice, half picturing two different objects. We then train a predictor similar to those used for the splits in the chopping scheme: feature selection with CMIM, and linear combination with a perceptron (see section 3.1), using the 30,000 features described above.

**Chopping:** The performance of the chopping approach is estimated for several numbers of splits (from 1 to 1024). For each split we select 50 objects from the training objects, and select at random 1,000 training images of these objects. We generate an arbitrary balanced binary labelling of these 50 objects and label the training images accordingly. We then

build a predictor by selecting $2,000$ features with the CMIM algorithm, and combine them with a perceptron (see section 3.1).

To compensate for the limitation of our conditional independence assumptions we allow to add a fixed bias to the log-odds ratio (1). This type of correction is common when using naive-Bayes type assumptions. Using the remaining training objects as validation set, we compute this bias so as to minimize the validation error. We insist that no objects of the test classes be used for training.

To improve the performance of the splits, we also test a "smart" version of the chopping for which each split is built in two steps. The first step is similar to what is described above. From that first step, we remove the 10 objects for which the labelling prediction has the highest error rate, and re-build the split with the 40 remaining objects. This get rid of problematic objects or inconsistent labelling (for instance trying to force two similar objects to be in different halves of the split).

### 4.3   Results

The experiments demonstrate the good performance of chopping when only one example is available. Its optimal error rate, obtained for the largest number of splits, is $7.41\%$ on the LaTeX symbol database and $11.42\%$ on the COIL-100 database. By contrast, a direct learning of the similarity (see section 4.2), reaches respectively $15.54\%$ and $18.1\%$ respectively with $8,192$ features.

On both databases, the classical multi-sample learning scheme requires 32 samples to reach the same level of performances ($10.51\%$ on the COIL-100 and $10.7\%$ on the LaTeX symbols).

The error curves (see figure 4) are all monotonic. There is no overfitting when the number of splits increases, which is consistent with the absence of global learning: splits are combined with an ad-hoc Bayesian rule, without optimizing a global functional, which generally also results in better robustness.

The smart splits (see section 4.2) achieve better performance initially but eventually reach the same error rates as the standard splits. There is no visible degradation of the asymptotic performance due to either a reduced independence between splits or a diminution of their separation power. However the computational cost is twice as high, since every predictor has to be built twice.

## 5   Conclusion

In this paper we have proposed an original approach to learning the appearance of an object from a single image. Our method relies on a large number of individual splits of the image space designed to keep together the images of any of the training objects. These splits are learned from a training set of examples and combined into a Bayesian framework to estimate the posterior probability for two images to show the same object.

This approach is very generic since it never makes the space of admissible perturbations explicit and relies on the generalization properties of the family of predictors. It can be applied to predict the similarity of two signals as soon as a family of binary predictors exists on the space of individual signals.

Since the learning is decomposed into the training of several splits independently, it can be easily parallelized. Also, because the combination rule is symmetric with respect to the splits, the learning can be incremental: splits can be added to the global rule progressively when they become available.

**Appendix: Proof of formula** (2). For the first factor, we have

$$P(L^1, L^2 \mid C_1 = C_2)$$

$$= \sum_{s^1, s^2} P(L^1, L^2 \mid C_1 = C_2, S^1 = s^1, S^2 = s^2) P(S^1 = s^1, S^2 = s^2 \mid C_1 = C_2)$$

$$= \sum_{s^1, s^2} P(L^1, L^2 \mid S^1 = s^1, S^2 = s^2) P(S^1 = s^1, S^2 = s^2 \mid C_1 = C_2)$$

$$= \sum_{s^1, s^2} \prod_i P(L_i^1 \mid S_i^1 = s_i^1) P(L_i^2 \mid S_i^2 = s_i^2) P((S_i^1, S_i^2) = (s_i^1, s_i^2) \mid C_1 = C_2)$$

$$= 2^{-N} \prod_i \left( P(L_i^1 \mid S_i^1 = 1) P(L_i^2 \mid S_i^2 = 1) + P(L_i^1 \mid S_i^1 = 0) P(L_i^2 \mid S_i^2 = 0) \right).$$

In the second equality, we have used that $L$ is independent of $C$ given $S$. In the third equality, we have used that the $(L_i^j)$ are independent given $S$. In the last equality, we have used the symmetry assumption on the distribution of $(S_1, S_2)$ given $C_1 = C_2$. Similarly,

$$P(L^1, L^2 \mid C_1 \neq C_2) = 4^{-N} \prod_i \sum_{s_1, s_2} P(L_i^1 \mid S_i^1 = s_1) P(L_i^2 \mid S_i^2 = s_2)$$

$$= 4^{-N} \prod_i P(L_i^1) P(L_i^2) \sum_{s_1, s_2} \frac{P(S_i^1 = s_1 \mid L_i^1) P(S_i^2 = s_2 \mid L_i^2)}{P(S_i^1 = s_1) P(S_i^2 = s_2)}$$

$$= 4^{-2N} \prod_i P(L_i^1) P(L_i^2),$$

since $P(S_i^j = s) \equiv \frac{1}{2}$ by the symmetry hypothesis. Taking the ratio of the two factors and using the latter property again leads to the conclusion.

## References

[1] Y. Bengio and M. Monperrus. Non-local manifold tangent learning. In *Advances in Neural Information Processing Systems 17*, pages 129–136. MIT press, 2005.

[2] T. Dietterich and G. Bakiri. Solving multiclass learning problems via error-correcting output codes. *Journal of Artificial Intelligence Research*, 2:263–286, 1995.

[3] A. Ferencz, E. Learned-Miller, and J. Malik. Learning hyper-features for visual identification. In *Advances in Neural Information Processing Systems 17*, pages 425–432. MIT Press, 2004.

[4] A. Ferencz, E. Learned-Miller, and J. Malik. Building a classification cascade for visual identification from one example. In *International Conference on Computer Vision (ICCV)*, 2005.

[5] M. Fink. Object classification from a single example utilizing class relevance metrics. In *Advances in Neural Information Processing Systems 17*, pages 449–456. MIT Press, 2005.

[6] F. Fleuret. Fast binary feature selection with conditional mutual information. *Journal of Machine Learning Research*, 5:1531–1555, November 2004.

[7] F. Li, R. Fergus, and P. Perona. A Bayesian approach to unsupervised one-shot learning of object categories. In *Proceedings of ICCV*, volume 2, page 1134, 2003.

[8] E. G. Miller, N. E. Matsakis, and P. A. Viola. Learning from one example through shared densities on transforms. In *Proceedings of the IEEE conference on Computer Vision and Pattern Recognition*, volume 1, pages 464–471, 2000.

[9] S. A. Nene, S. K. Nayar, and H. Murase. Columbia Object Image Library (COIL-100). Technical Report CUCS-006-96, Columbia University, 1996.

[10] T. Sejnowski and C. Rosenberg. Parallel networks that learn to pronounce english text. *Journal of Complex Systems*, 1:145–168, 1987.

[11] P. Simard, Y. Le Cun, and J. Denker. Efficient pattern recognition using a new transformation distance. In S. Hanson, J. Cowan, and C. Giles, editors, *Advances in Neural Information Processing Systems 5*, pages 50–68. Morgan Kaufmann, 1993.

[12] S. Thrun and L. Pratt, editors. *Learning to learn*. Kluwer, 1997.
